# The Capacity of a Bump

**Gary William Flake***
Institute for Advance Computer Studies
University of Maryland
College Park, MD 20742

## Abstract

Recently, several researchers have reported encouraging experimental results when using Gaussian or bump-like activation functions in multilayer perceptrons. Networks of this type usually require fewer hidden layers and units and often learn much faster than typical sigmoidal networks. To explain these results we consider a *hyper-ridge* network, which is a simple perceptron with no hidden units and a ridge activation function. If we are interested in partitioning $p$ points in $d$ dimensions into two classes then in the limit as $d$ approaches infinity the capacity of a hyper-ridge and a perceptron is identical. However, we show that for $p \gg d$, which is the usual case in practice, the ratio of hyper-ridge to perceptron dichotomies approaches $p/2(d+1)$.

## 1 Introduction

A *hyper-ridge* network is a simple perceptron with no hidden units and a ridge activation function. With one output this is conveniently described as $y = g(h) = g(\mathbf{w} \cdot \mathbf{x} - b)$ where $g(h) = \text{sgn}(1 - h^2)$. Instead of dividing an input-space into two classes with a single hyperplane, a hyper-ridge network uses two parallel hyperplanes. All points in the interior of the hyperplanes form one class, while all exterior points form another. For more information on hyper-ridges, learning algorithms, and convergence issues the curious reader should consult [3].

We wouldn't go so far as to suggest that anyone actually use a hyper-ridge for a real-world problem, but it is interesting to note that a hyper-ridge can represent linear inseparable mappings such as XOR, NEGATE, SYMMETRY, and COUNT($m$) [2, 3]. Moreover, hyper-ridges are very similar to multilayer perceptrons with bump-like activation functions, such as a Gaussian, in the way the input space is partitioned. Several researchers [6, 2, 3, 5] have independently found that Gaussian units offer many advantages over sigmoidal units.

*Current address: Adaptive Information and Signal Processing Department, Siemens Corporate Research, 755 College Road East, Princeton, NJ 08540. Email: flake@scr.siemens.com

In this paper we derive the capacity of a hyper-ridge network. Our first result is that hyper-ridges and simple perceptrons are equivalent in the limit as the input dimension size approaches infinity. However, when the number of patterns is far greater than the input dimension (as is the usual case) the ratio of hyper-ridge to perceptron dichotomies approaches $p/2(d+1)$, giving some evidence that bump-like activation functions offer an advantage over the more traditional sigmoid.

The rest of this paper is divided into three more sections. In Section 2 we derive the number of dichotomies for a hyper-ridge network. The capacities for hyper-ridges and simple perceptrons are compared in Section 3. Finally, in Section 4 we give our conclusions.

## 2  The Representation Power of a Hyper-Ridge

Suppose we have $p$ patterns in the pattern-space, $\Re^d$, where $d$ is the number of inputs of our neural network. A *dichotomy* is a classification of all of the points into two distinct sets. Clearly, there are at most $2^p$ dichotomies that exist. We are concerned with the number of dichotomies that a single hyper-ridge node can represent. Let the number of dichotomies of $p$ patterns in $d$ dimensions be denoted as $D(p, d)$.

For the case of $D(1, d)$, when $p = 1$ there are always two and only two dichotomies since one can trivially include the single point or no points. Thus, $D(1, d) = 2$.

For the case of $D(p, 1)$, all of the points are constrained to fall on a line. From this set pick two points, say $x_a$ and $x_b$. It is always possible to place a ridge function such that all points between $x_a$ and $x_b$ (inclusive of the end points) are included in one set, and all other points are excluded. Thus, there are $p$ dichotomies consisting of a single point, $p - 1$ dichotomies consisting of two points, $p - 2$ dichotomies consisting of three points, and so on. No other dichotomies besides the empty set are possible. The number of possible hyper-ridge dichotomies in one dimension can now be expressed as

$$D(p, 1) = \sum_{i=1}^{p} i + 1 = \frac{1}{2}p(p + 1) + 1,  \tag{1}$$

with the extra dichotomy coming from the empty set.

To derive the general form of the recurrence relationship, we would have to resort to techniques similar to those used by Cover [1], Nilsson [7], and Gardner [4]. Because of space considerations, we do not give the full derivation of the general form of the recurrence relationship in this paper, but instead cite the complete derivation given in [3]. The short version of the story is that the general form of the recurrence relationship for hyper-ridge dichotomies is identical to the equivalent expression for simple perceptrons:

$$D(p, d) = D(p - 1, d) + D(p - 1, d - 1).  \tag{2}$$

All differences between the capacity of hyper-ridges and simple perceptrons are, therefore, a consequence of the different base cases for the recurrence expression.

To get Equation 2 into closed form, we first expand $D(p, d)$ a total of $p$ times, yielding

$$D(p, d) = \sum_{i=0}^{p-1} \binom{p-1}{i} D(1, d - i).  \tag{3}$$

For Equation 3 it is possible for the second term of $D(1, d - 1)$ to become zero or negative. Taking the two identities $D(p, 0) = p + 1$ and $D(p, -1) = 1$ are the only choices that are consistent with the recurrent relationship expressed in Equation 2. With this in mind, there are three separate cases that we need to be concerned with: $p < d + 2$, $p = d + 2$, and

$p > d + 2$. When $p < d + 2$

$$D(p, d) = \sum_{i=0}^{p-1} \binom{p-1}{i} D(1, d-i) = 2 \sum_{i=0}^{p-1} \binom{p-1}{i} = 2^p, \tag{4}$$

since all of the second terms in $D(1, d - i)$ are always greater or equal to zero. When $p = d + 2$, the last term in $D(1, d - i)$, in the summation, will be equal to $-1$. Thus we can expand Equation 3 in this case to

$$
\begin{aligned}
D(p, d) &= \sum_{i=0}^{p-1} \binom{p-1}{i} D(1, d-i) = \sum_{i=0}^{p-1} \binom{p-1}{i} D(1, p-2-i) \\
&= \sum_{i=0}^{p-2} \binom{p-1}{i} D(1, p-2-i) + 1 = 2 \sum_{i=0}^{p-2} \binom{p-1}{i} + 1 \\
&= 2(2^{p-1} - 1) + 1 = 2^p - 1.
\end{aligned} \tag{5}
$$

Finally, when $p > d + 2$, some of the last terms in $D(1, d - i)$ are always negative. We can disregard all $d - i < -1$, taking $D(1, d - i)$ equal to zero in these cases (which is consistent with the recurrence relationship),

$$
\begin{aligned}
D(p, d) &= \sum_{i=0}^{p-1} \binom{p-1}{i} D(1, d-i) = \sum_{i=0}^{d+1} \binom{p-1}{i} D(1, d-i) \\
&= \sum_{i=0}^{d} \binom{p-1}{i} D(1, d-i) + \binom{p-1}{d+1} = 2 \sum_{i=0}^{d} \binom{p-1}{i} + \binom{p-1}{d+1}.
\end{aligned} \tag{6}
$$

Combining Equations 4, 5, and 6 gives

$$
D(p, d) = \begin{cases}
2 \sum_{i=0}^{d} \binom{p-1}{i} + \binom{p-1}{d+1} & \text{for } p > d + 2 \\
2^p - 1 & \text{for } p = d + 2 \\
2^p & \text{for } p < d + 2
\end{cases} \tag{7}
$$

## 3   Comparing Representation Power

Cover [1], Nilsson [7], and Gardner [4] have all shown that $D(p, d)$ for simple perceptrons obeys the rule

$$
D(p, d) = \begin{cases}
2 \sum_{i=0}^{d} \binom{p-1}{i} & \text{for } p > d + 2 \\
2^p - 2 & \text{for } p = d + 2 \\
2^p & \text{for } p < d + 2
\end{cases} \tag{8}
$$

The interesting case is when $p > d + 2$, since that is where Equations 7 and 8 differ the most. Moreover, problems are more difficult when the number of training patterns greatly exceeds the number of trainable weights in a neural network.

Let $D_h(p, d)$ and $D_p(p, d)$ denote the number of dichotomies possible for hyper-ridge networks and simple perceptrons, respectively. Additionally, Let $C_h$, and $C_p$ denote the

respective capacities. We should expect both $D_h(p, d)/2^p$ and $D_p(p, d)/2^p$ to be at or around 1 for small values of $p/(d + 1)$. At some point, for large $p/(d + 1)$, the $2^p$ term should dominate, making the ratio go to zero. The capacity of a network can loosely be defined as the value $p/(d + 1)$ such that $D(p, d)/2^p = \frac{1}{2}$. This is more rigorously defined as

$$C = \left\{ c : \lim_{d \to \infty} \frac{D(c(d + 1), d)}{2^{c(d+1)}} = \frac{1}{2} \right\},$$

which is the point in which the transition occurs in the limit as the input dimension goes to infinity.

Figures 1, 2, and 3 illustrate and compare $C_p$ and $C_h$ at different stages. In Figure 1 the capacities are illustrated for perceptrons and hyper-ridges, respectively, by plotting $D(p, d)/2^p$ versus $p/(d + 1)$ for various values of $d$. On par with our intuition, the ratio $D(p, d)/2^p$ equals 1 for small values of $p/(d + 1)$ but decreases to zero as $p(d + 1)$ increases. Figure 2 and the left diagram of Figure 3 plot $D(p, d)/2^p$ versus $p/(d + 1)$ for perceptron and hyper-ridges, side by side, with values of $d = 5, 20$, and 100. As $d$ increases, the two curves become more similar. This fact is further illustrated in the right diagram of Figure 3 where the plot is of $D_h(p, d)/D_p(p, d)$ versus $p$ for various values of $d$. The ratio clearly approaches 1 as $d$ increases, but there is significant difference for smaller values of $d$.

The differences between $D_p$ and $D_h$ can be more explicitly quantified by noting that

$$D_h(p, d) = D_p(p, d) + \binom{p - 1}{d + 1}$$

for $p > d + 2$. This difference clearly shows up in in the plots comparing the two capacities. We will now show that the capacities are identical in the limit as $d$ approaches infinity. To do this, we will prove that the capacity curves for both hyper-ridges and perceptrons crosses $\frac{1}{2}$ at $p/(d + 1) = 2$. This fact is already widely known for perceptrons. Because of space limitations we will handwave our way through lemma and corollary proofs. The curious reader should consult [3] for the complete proofs.

**Lemma 3.1**

$$\lim_{n \to \infty} \frac{1}{2^{2n}} \binom{2n}{n} = 0.$$

**Short Proof** Since $n$ approaches infinity, we can use Stirling's formula as an approximation of the factorials.

□

**Corollary 3.2** *For all positive integer constants, a, b, and c,*

$$\lim_{n \to \infty} \frac{1}{2^{2n+a}} \binom{2n + b}{n + c} = 0.$$

**Short Proof** When adding the constants $b$ and $c$ to the combination, the whole combination can always be represented as $\text{comb}(2n, n) \cdot \gamma$, where $\gamma$ is some multiplicative constant. Such a constant can always be factored out of the limit. Additionally, large values of $a$ only increase the growth rate of the denominator.

□

**Lemma 3.3** *For $p/(d + 1) = 2$, $\lim_{d \to \infty} D_p(p, d)/2^p = \frac{1}{2}$.*

**Short Proof** Consult any of Cover [1], Nilsson [7], or Gardner [4] for full proof.

□

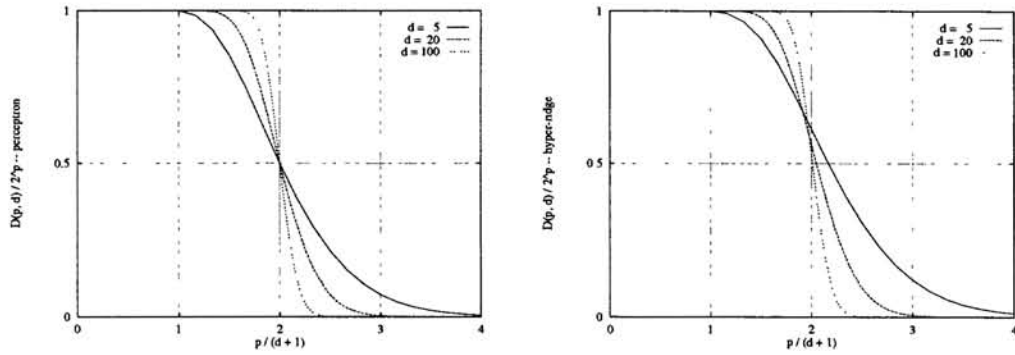

Figure 1: On the left, $D_p(p, d)/2^p$ versus $p/(d + 1)$, and on the right, $D_h(p, d)/2^p$ versus $p/(d + 1)$ for various values of $d$. Notice that for perceptrons the curve always passes through $\frac{1}{2}$ at $p/(d + 1) = 2$. For hyper-ridges, the point where the curve passes through $\frac{1}{2}$ decreases as $d$ increases.

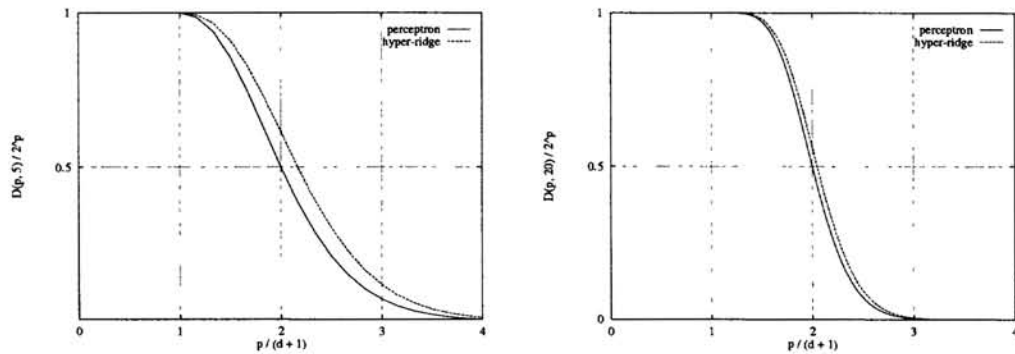

Figure 2: On the left, capacity comparison for $d = 5$. There is considerable difference for small values of $d$, especially when one considers that the capacities are normalized by $2^p$. On the right, comparison for $d = 20$. The difference between the two capacities is much more subtle now that $d$ is fairly large.

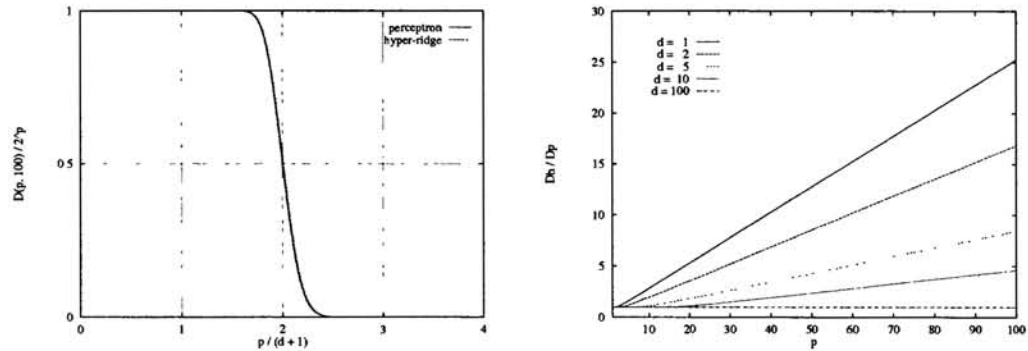

Figure 3: On the left, capacity comparison for $d = 100$. For this value of $d$, the capacities are visibly indistinguishable. On the right, $D_h(p, d)/D_p(p, d)$ versus $p$ for various values of $d$. For small values of $d$ the capacity of a hyper-ridge is much greater than a perceptron. As $d$ grows, the ratio asymptotically approaches 1.

**Theorem 3.4** *For $p/(d + 1) = 2$,*

$$\lim_{d \to \infty} \frac{D_h(p, d)}{2^p} = \frac{1}{2}.$$

**Proof** Taking advantage of the relationship between perceptron dichotomies and hyper-ridge dichotomies allows us to expand $D_h(p, d)$,

$$\lim_{d \to \infty} \frac{D_h(p, d)}{2^p} = \lim_{d \to \infty} \frac{D_p(p, d)}{2^p} + \lim_{d \to \infty} \binom{p - 1}{d + 1}.$$

By Lemma 3.3, and substituting $2(d + 1)$ for $p$, we get:

$$\frac{1}{2} + \lim_{d \to \infty} \binom{2d + 1}{d + 1}.$$

Finally, by Corollary 3.2 the right limit vanishes leaving us with $\frac{1}{2}$.
□

Superficially, Theorem 3.4 would seem to indicate that there is no difference between the representation power of a perceptron and a hyper-ridge network. However, since this result is only valid in the limit as the number of inputs goes to infinity, it would be interesting to know the exact relationship between $D_p(d, p)$ and $D_h(d, p)$ for finite values of $d$.

In the right diagram of Figure 3 values of $D_p(d, p)/D_h(d, p)$ are plotted against various values of $p$. The figure is slightly misleading since the ratio appears to be linear in $p$, when, in fact, the ratio is only approximately linear in $p$. If we normalize the ratio by $\frac{1}{p}$ and recompute the ratio in the limit as $p$ approaches infinity the ratio becomes linear in $d$. Theorem 3.5 establishes this rigorously.

**Theorem 3.5**

$$\lim_{p \to \infty} \frac{1}{p} \frac{D_h(d, p)}{D_p(d, p)} = \frac{1}{2(d + 1)}$$

**Proof** First, note that we can simplify the left hand side of the expression to

$$\lim_{p \to \infty} \frac{1}{p} \frac{D_h(d, p)}{D_p(d, p)} = \lim_{p \to \infty} \frac{1}{p} \frac{D_p(d, p) + \binom{p - 1}{d + 1}}{D_p(d, p)} = \lim_{p \to \infty} \frac{1}{p} \frac{\binom{p - 1}{d + 1}}{D_p(d, p)} \quad (9)$$

In the next step, we will invert Equation 9, making it easier to work with. We need to show that the new expression is equal to $2(d + 1)$.

$$\lim_{p \to \infty} p \frac{D_p(d, p)}{\binom{p - 1}{d + 1}} = \lim_{p \to \infty} 2p \frac{\sum_{i=0}^{d} \binom{p - 1}{i}}{\binom{p - 1}{d + 1}} =$$

$$\lim_{p \to \infty} 2p \sum_{i=0}^{d} \frac{(p - 1)!}{i!(p - i - 1)!} \frac{(d + 1)!(p - d - 2)!}{(p - 1)!} = \lim_{p \to \infty} 2p \sum_{i=0}^{d} \frac{(d + 1)!}{i!} \frac{(p - d - 2)!}{(p - i - 1)!} =$$

$$\lim_{p \to \infty} \frac{p}{(p - 1 - d)} 2(d + 1) \sum_{i=0}^{d} \frac{d!}{i!} \frac{(p - d - 1)!}{(p - i - 1)!} = \lim_{p \to \infty} 2(d + 1) \sum_{i=0}^{d} \frac{d!}{i!} \frac{(p - d - 1)!}{(p - i - 1)!} \quad (10)$$

In Equation 10, the summation can be reduced to 1 since

$$\lim_{p \to \infty} \frac{d!}{i!} \frac{(p - d - 1)!}{(p - i - 1)!} = \begin{cases} 0 & \text{when } 0 \leq i < d \\ 1 & \text{when } i = d \end{cases}.$$

Thus, Equation 10 is equal to $2(d + 1)$, which proves the theorem.

□

Theorem 3.5 is valid only in the case when $p \gg d$, which is typically true in interesting classification problems. The result of the theorem gives us a good estimate of how many more dichotomies are computable with a hyper-ridge network when compared to a simple perceptron. When $p \gg d$ the equation

$$\frac{D_h(d, p)}{D_p(d, p)} \simeq \frac{p}{2(d + 1)} \tag{11}$$

is an accurate estimate of the difference between the capacities of the two architectures. For example, taking $d = 4$ and $p = 60$ and applying the values to Equation 11 yields the ratio of 6, which should be interpreted as meaning that one could store six times the number of mappings in a hyper-ridge network than one could in a simple perceptron. Moreover, Equation 11 is in agreement with the right diagram of Figure 3 for all values of $p \gg d$.

## 4   Conclusion

An interesting footnote to this work is that the VC dimension [8] of a hyper-ridge network is identical to a simple perceptron, namely $d$. However, the real difference between perceptrons and hyper-ridges is more noticeable in practice, especially when one considers that linear inseparable problems are representable by hyper-ridges.

We also know that there is no such thing as a free lunch and that generalization is sure to suffer in just the cases when representation power is increased. Yet given all of the comparisons between MLPs and radial basis functions (RBFs) we find it encouraging that there may be a class of approximators that is a compromise between the local nature of RBFs and the global structure of MLPs.

## References

[1] T.M. Cover. Geometrical and statistical properties of systems of linear inequalities with applications in pattern recognition. *IEEE Transactions on Electronic Computers*, 14:326–334, 1965.

[2] M.R.W. Dawson and D.P. Schopflocher. Modifying the generalized delta rule to train networks of non-monotonic processors for pattern classification. *Connection Science*, 4(1), 1992.

[3] G. W. Flake. *Nonmonotonic Activation Functions in Multilayer Perceptrons*. PhD thesis, University of Maryland, College Park, MD, December 1993.

[4] E. Gardner. Maximum storage capacity in neural networks. *Europhysics Letters*, 4:481–485, 1987.

[5] F. Girosi, M. Jones, and T. Poggio. Priors, stabilizers and basis functions: from regularization to radial, tensor and additive splines. Technical Report A.I. Memo No. 1430, C.B.C.L. Paper No. 75, MIT AI Laboratory, 1993.

[6] E. Hartman and J. D. Keeler. Predicting the future: Advanges of semilocal units. *Neural Computation*, 3:566–578, 1991.

[7] N.J. Nilsson. *Learning Machines: Foundations of Trainable Pattern Classifying Systems*. McGraw-Hill, New York, 1965.

[8] V.N. Vapnik and A.Y. Chervonenkis. On the uniform convergence of relative frequencies of events to their probabilities. *Theory of Probability and Its Applications*, 16:264–280, 1971.
